# Sparse Approximate Manifolds for Differential Geometric MCMC

**Ben Calderhead**[*]
CoMPLEX
University College London
London, WC1E 6BT, UK
b.calderhead@ucl.ac.uk

**Mátyás A. Sustik**
Department of Computer Sciences
University of Texas at Austin
Austin, TX 78712, USA
sustik@cs.utexas.edu

## Abstract

One of the enduring challenges in Markov chain Monte Carlo methodology is the development of proposal mechanisms to make moves distant from the current point, that are accepted with high probability and at low computational cost. The recent introduction of locally adaptive MCMC methods based on the natural underlying Riemannian geometry of such models goes some way to alleviating these problems for certain classes of models for which the metric tensor is analytically tractable, however computational efficiency is not assured due to the necessity of potentially high-dimensional matrix operations at each iteration.

In this paper we firstly investigate a sampling-based approach for approximating the metric tensor and suggest a valid MCMC algorithm that extends the applicability of Riemannian Manifold MCMC methods to statistical models that do not admit an analytically computable metric tensor. Secondly, we show how the approximation scheme we consider naturally motivates the use of $\ell_1$ regularisation to improve estimates and obtain a sparse approximate inverse of the metric, which enables stable and sparse approximations of the local geometry to be made. We demonstrate the application of this algorithm for inferring the parameters of a realistic system of ordinary differential equations using a biologically motivated robust Student-t error model, for which the Expected Fisher Information is analytically intractable.

## 1  Introduction

The use of Markov chain Monte Carlo methods can be extremely challenging in many modern day applications. This difficulty arises from the more frequent use of complex and nonlinear statistical models that induce strong correlation structures in their often high-dimensional parameter spaces. The exact structure of the target distribution is generally not known in advance and local correlation structure between different parameters may vary across the space, particularly as the chain moves from the transient phase, exploring areas of negligible probability mass, to the stationary phase exploring higher density regions [1].

Constructing a Markov chain that adapts to the target distribution while still drawing samples from the correct stationary distribution is challenging, although much research over the last 15 years has resulted in a variety of approaches and theoretical results. Adaptive MCMC for example, allows for *global* adaptation based on the partial or full history of a chain; this breaks its Markov property, although it has been shown that subject to some technical conditions [2,3] the resulting chain will still converge to the desired stationary distribution. Most recently, advances in Riemannian Manifold MCMC allow *locally* changing, position specific proposals to be made based on the underlying

---

[*]http://www.2020science.net/people/ben-calderhead

geometry of the target distribution [1]. This directly takes into account the changing sensitivities of the model for different parameter values and enables very efficient inference over a number of popular statistical models. It is useful for inference over large numbers of strongly covarying parameters, however this methodology is still not suitable for all statistical models; in its current form it is only applicable to models that admit an analytic expression for the metric tensor. In practice, there are many commonly used models for which the Expected Fisher Information is not analytically tractable, such as when a robust Student-t error model is employed to construct the likelihood.

In this paper we propose the use of a locally adaptive MCMC algorithm that approximates the local Riemannian geometry at each point in the target space. This extends the applicability of Riemannian Manifold MCMC to a much wider class of statistical models than at present. In particular, we do so by estimating the covariance structure of the tangent vectors at a point on the Riemannian manifold induced by the statistical model. Considering this geometric problem as one of inverse covariance estimation naturally leads us to the use of an $\ell_1$ regularised maximum likelihood estimator. This approximate inverse approach allows the required geometry to be estimated with few samples, enabling good proposals for the Markov chain while inducing a natural sparsity in the inverse metric tensor that reduces the associated computational cost.

We first give a brief characterisation of current adaptive approaches to MCMC, making a distinction between locally and globally adaptive methods, since these two approaches have very different requirements in terms of proving convergence to the stationary distribution. We then discuss the use of geometry in MCMC and the interpretation of such methods as being locally adaptive, before giving the necessary background on Riemannian geometry and MCMC algorithms defined on induced Riemannian manifolds. We focus on the manifold MALA sampler, which is derived from a Langevin diffusion process that takes into account local non-Euclidean geometry, and we discuss simplifications that may be made for computational efficiency. Finally we present a valid MCMC algorithm that estimates the Riemannian geometry at each iteration based on covariance estimates of random vectors tangent to the manifold at the chain's current point. We demonstrate the use of $\ell_1$ regularisation to calculate sparse approximate inverses of the metric tensor and investigate the sampling properties of the algorithm on an extremely challenging statistical model for which the Expected Fisher Information is analytically intractable.

## 2  Background

We wish to sample from some arbitrary target density $\pi(\mathbf{x})$ defined on a continuous state space $\mathbf{X}^D$, which may be high-dimensional. We may define a Markov chain that converges to the correct stationary distribution in the usual manner by proposing a new position $\mathbf{x}^*$ from the current position $\mathbf{x}_n$ via some fixed proposal distribution $q(\mathbf{x}^*|\mathbf{x}_n)$; we accept the new move setting $\mathbf{x}_{n+1} = \mathbf{x}^*$ with probability $\alpha(\mathbf{x}^*|\mathbf{x}_n) = \min(\frac{\pi(\mathbf{x}^*)}{\pi(\mathbf{x}_n)}\frac{q(\mathbf{x}_n|\mathbf{x}^*)}{q(\mathbf{x}^*|\mathbf{x}_n)}, 1)$ and set $\mathbf{x}_{n+1} = \mathbf{x}_n$ otherwise. In a Bayesian context, we will often have a posterior distribution as our target $\pi(\mathbf{x}) = p(\boldsymbol{\theta}|\mathbf{y})$, where $\mathbf{y}$ is the data and $\boldsymbol{\theta}$ are the parameters of a statistical model. The choice of proposal distribution is the critical factor in determining how efficiently the Markov chain can explore the space and whether new moves will be accepted with high probability and be sufficiently far from the current point to keep autocorrelation of the samples to a minimum. There is a lot of flexibility in the choice of proposal distribution, in that it may depend on the current point in a deterministic manner.

We note that Adaptive MCMC approaches attempt to change their proposal mechanism throughout the running of the algorithm, and for the purpose of proving convergence to the stationary distribution it is useful to categorise them as follows; *locally* adaptive MCMC methods make proposals based only on the current position of the chain, whereas *globally* adaptive MCMC methods use previously collected samples in the chain's history to generate a new proposal mechanism. This is an important distinction since globally adaptive methods lose their Markov property and convergence to the stationary distribution must be proven in an alternative manner. It has been shown that such chains may still be usefully employed as long as they satisfy some technical conditions, namely diminishing adaptation and bounded convergence [2]. In practice these algorithms represent a step towards MCMC as a "black box" method and may be very useful for sampling from target distributions for which there is no derivative or higher order geometric information available, however there are simple examples of standard Adaptive MCMC methods requiring hundreds of thousands of iterations in higher dimensions before adapting to a suitable proposal distribution [3]. In addi-

tion, if there is more information about the target density available, then there seems little point in trying to guess the geometric structure when it may be calculated directly. In this paper we focus on *locally* adaptive methods that employ proposals constructed deterministically from information at the current position of the Markov chain.

## 2.1 Locally Adaptive MCMC

Many geometric-based MCMC methods may be categorised as being locally adaptive. When the derivative of the target density is available, MCMC methods such as the Metropolis-adjusted Langevin Algorithm (MALA) [4] allow local adaptation based on the geometry at the current point, but unlike globally adaptive MCMC, they retain their Markovian property and therefore converge to the correct stationary distribution using a standard Metropolis-Hastings step and without the need to satisfy further technical conditions.

In general, we can define position-specific proposal densities based on deterministic functions that depend only on the current point. This idea has been previously employed to develop approaches for sampling multimodal distributions whereby large initial jumps followed by deterministic optimisation functions were used to create mode-jumping proposal mechanisms [5]. In some instances, the use of first order geometric information may drastically speed up the convergence to a stationary distribution, however in other cases such algorithms exhibit very slow convergence, due to the gradients not being isotropic in magnitude [6]; in practice gradients may vary greatly in different directions and the rate of exploration of the target density may in addition be dependent on the problem-specific choice of parameterisation [1]. Methods using the standard gradient implicitly assume that the slope in each direction is approximately constant over a small distance, when in fact these gradients may rapidly change over short distances. Incorporating higher order geometry often helps although at an increased computational cost.

A number of Hessian-based MCMC methods have been proposed as a solution [7]. While such approaches have been shown to work very well for selected problems there are a number of problems with this use of geometry; ad hoc methods are often necessary to deal with the fact that the Hessian might not be everywhere positive-definite, and second derivatives can be challenging and costly to compute. We can also exploit higher order information through the use of Riemannian geometry. Using a metric tensor instead of a Hessian matrix lends us nice properties such as invariance to reparameterisation of our statistical model, and positive-definiteness is also assured. Riemannian geometry has been useful in a variety of other machine learning and statistical contexts [8] however the limiting factor is usually analytic or computational tractability.

## 3 Differential Geometric MCMC

During the 1940s, Jeffreys and Rao demonstrated that the Expected Fisher Information has the same properties as a metric tensor and indeed induces a natural Riemannian structure for a statistical model [11, 10], providing a fascinating link between statistics and differential geometry. Much work has been done since then elucidating the relationship between statistics and Riemannian geometry, in particular examining geometric concepts such as distance, curvature and geodesics on statistical manifolds, within a field that has become known as Information Geometry [6]. We first provide an overview of Riemannian geometry and MCMC algorithms defined on Riemannian manifolds. We then describe a sampling scheme that allows the local geometry to be *estimated* at each iteration for statistical models that do not admit an analytically tractable metric tensor.

## 3.1 Riemannian Geometry

Informally, a manifold is an $n$-dimensional space that is locally Euclidean; it is locally equivalent to $\mathbb{R}^n$ via some smooth transformation. At each point $\boldsymbol{\theta} \in \mathbb{R}^n$ on a Riemannian manifold $M$ there exists a tangent space, which we denote as $T_{\boldsymbol{\theta}}M$. We can think of this as a linear approximation to the Riemannian manifold at the point $\boldsymbol{\theta}$ and this is simply a standard vector space, whose origin is the current point on the manifold and whose vectors are tangent to this point. The vector space $T_{\boldsymbol{\theta}}M$ is spanned by the differential operators $\left[\frac{\partial}{\partial\theta_1}, \ldots, \frac{\partial}{\partial\theta_n}\right]$, which act on functions defining paths on the underlying manifold [9]. In the context of MCMC we can consider the target density as the

log-likelihood of a statistical model given some data, such that at a particular point $\boldsymbol{\theta}$, the derivatives of the log-likelihood are tangent to the manifold and these are just the score vectors at $\boldsymbol{\theta}$, $\nabla_{\boldsymbol{\theta}}\mathcal{L} = \left[\frac{\partial}{\partial\theta_1}, \ldots, \frac{\partial}{\partial\theta_n}\right]$. The tangent space at each point $\boldsymbol{\theta}$ arises when we equip a differentiable manifold with an inner product at each point, which we can use to measure distance and angles between vectors. This inner product is defined in terms of a metric tensor, $G_{\boldsymbol{\theta}}$, which defines a basis on each tangent space $T_{\boldsymbol{\theta}}M$. The tangent space is therefore a linear approximation of the manifold at a given point and it has the same dimensionality. A natural inner product for this vector space is given by the covariance of the basis score vectors, since the covariance function satisfies the same properties as a metric tensor, namely symmetry, bilinearity and positive-definiteness [9]. This inner product then turns out to be equivalent to the Expected Fisher Information, following from the fact that the expectation of the score is zero, with the $[i, j]^{th}$ component of the tensor given by

$$G_{i,j} = \text{Cov}\left(\frac{\partial\mathcal{L}}{\partial\theta^i}, \frac{\partial\mathcal{L}}{\partial\theta^j}\right) = E_{p(\mathbf{x}|\boldsymbol{\theta})}\left(\frac{\partial\mathcal{L}}{\partial\theta^i}^T \frac{\partial\mathcal{L}}{\partial\theta^j}\right) = -E_{p(\mathbf{x}|\boldsymbol{\theta})}\left(\frac{\partial^2\mathcal{L}}{\partial\theta^i\partial\theta^j}\right) \qquad (1)$$

Each tangent vector, $\mathbf{t}_1 \in T_{\boldsymbol{\theta}}M$, at a point on the manifold, $\boldsymbol{\theta} \in M$, has a length $||\mathbf{t}_1|| \in \mathbb{R}^+$, whose square is given by the inner product, such that $||\mathbf{t}_1||^2_{G_{\boldsymbol{\theta}}} = \langle\mathbf{t}_1, \mathbf{t}_1\rangle_{\boldsymbol{\theta}} = \mathbf{t}_1^T G_{\boldsymbol{\theta}}\mathbf{t}_1$. This squared distance is known as the first fundamental form in Riemannian geometry [9], is invariant to reparameterisations of the coordinates, and importantly for MCMC provides a local measure of distance that takes into account the local 2nd order sensitivity of the statistical model. We note that when the metric tensor is constant for all values of $\boldsymbol{\theta}$ then the Riemannian manifold is equivalent to a vector space with constant inner product; further, if the metric tensor is an identity matrix then the manifold simply becomes a Euclidean space.

## 3.2 Manifold MCMC

We consider the manifold version of the MALA sampling algorithm, which proposes moves based on a stochastic differential equation defining a Langevin diffusion [4]. It turns out we can also define such a diffusion on a Riemannian manifold [12], and so in a similar manner we can derive a sampling algorithm that takes the underlying geometric structure into account when making proposals. It is based on the Laplace-Beltrami operator, which simply measures the divergence of a vector field on a manifold. The stochastic differential equation defining the Langevin diffusion on a Riemannian manifold is $d\boldsymbol{\theta}(t) = \frac{1}{2}\tilde{\nabla}_{\boldsymbol{\theta}}\mathcal{L}(\boldsymbol{\theta}(t))dt + d\tilde{\mathbf{b}}(t)$, where the natural gradient [6] is the gradient of a function transformed into the tangent space at the current point by a linear transformation using the basis defined by the metric tensor, such that $\tilde{\nabla}_{\boldsymbol{\theta}}\mathcal{L}(\boldsymbol{\theta}(t)) = G^{-1}(\boldsymbol{\theta}(t))\nabla_{\boldsymbol{\theta}}\mathcal{L}(\boldsymbol{\theta}(t))$, and the Brownian motion on the Riemannian manifold is defined as

$$d\tilde{\mathbf{b}}_i(t) = |G(\boldsymbol{\theta}(t))|^{-\frac{1}{2}}\sum_{j=1}^{D}\frac{\partial}{\partial\boldsymbol{\theta}_j}(G^{-1}(\boldsymbol{\theta}(t))_{ij}|G(\boldsymbol{\theta}(t))|^{\frac{1}{2}})dt + \left(\sqrt{G^{-1}(\boldsymbol{\theta}(t))}d\mathbf{b}(t)\right)_i \qquad (2)$$

The first part of the right hand side of Equation 2 represents the 1st order terms of the Laplace-Beltrami operator and these relate to the local curvature of the manifold, reducing to zero if the metric is everywhere constant. The second term on the right hand side provides a position specific linear transformation of the Brownian motion $\mathbf{b}(t)$ based on the local metric. Employing a first order Euler integrator, the discrete form of the Langevin diffusion on a Riemannian manifold follows as

$$
\begin{aligned}
\boldsymbol{\theta}_i^{n+1} = {}& \boldsymbol{\theta}_i^n + \frac{\epsilon^2}{2}(G^{-1}(\boldsymbol{\theta}^n)\nabla_{\boldsymbol{\theta}}\mathcal{L}(\boldsymbol{\theta}^n))_i - \epsilon^2\sum_{j=1}^{D}\left(G^{-1}(\boldsymbol{\theta}^n)\frac{\partial G(\boldsymbol{\theta}^n)}{\partial\boldsymbol{\theta}_j}G^{-1}(\boldsymbol{\theta}^n)\right)_{ij} \qquad (3)\\
& + \frac{\epsilon^2}{2}\sum_{j=1}^{D}\left(G^{-1}(\boldsymbol{\theta}^n)\right)_{ij}Tr\left(G^{-1}(\boldsymbol{\theta}^n)\frac{\partial G(\boldsymbol{\theta}^n)}{\partial\boldsymbol{\theta}_j}\right) + \left(\epsilon\sqrt{G^{-1}(\boldsymbol{\theta}^n)}\mathbf{z}^n\right)_i\\
= {}& \boldsymbol{\mu}(\boldsymbol{\theta}^n, \epsilon)_i + \left(\epsilon\sqrt{G^{-1}(\boldsymbol{\theta}^n)}\mathbf{z}^n\right)_i
\end{aligned}
$$

which defines a proposal mechanism with density $q(\boldsymbol{\theta}^*|\boldsymbol{\theta}^n) = \mathcal{N}(\boldsymbol{\theta}^*|\boldsymbol{\mu}(\boldsymbol{\theta}^n, \epsilon), \epsilon^2 G^{-1}(\boldsymbol{\theta}^n))$ and acceptance probability $\min\{1, p(\boldsymbol{\theta}^*)q(\boldsymbol{\theta}^n|\boldsymbol{\theta}^*)/p(\boldsymbol{\theta}^n)q(\boldsymbol{\theta}^*|\boldsymbol{\theta}^n)\}$ to ensure convergence to the invariant density $p(\boldsymbol{\theta})$. We note that this deterministically defines a position-specific proposal distribution at each point on the manifold; we may categorise this as another locally adaptive MCMC method and convergence to the invariant density follows from using the standard Metropolis-Hastings ratio.

It may be computationally expensive to calculate the 3rd order derivatives needed for working out the rate of change of the metric tensor, and so an obvious approximation is to assume these derivatives are zero for each step. In other words, for each step we can assume that the metric is locally constant. Of course even if the curvature of the manifold is not constant, this simplified proposal mechanism still defines a correct MCMC method which converges to the target measure, as we accept or reject moves using a Metropolis-Hastings ratio. This is equivalent to a position-specific pre-conditioned MALA proposal, where the pre-conditioning is dependent on the current parameter values

$$\boldsymbol{\theta}^{n+1} = \boldsymbol{\theta}^n + \frac{\epsilon^2}{2}G^{-1}(\boldsymbol{\theta}^n)\nabla_{\boldsymbol{\theta}}\mathcal{L}(\boldsymbol{\theta}^n) + \epsilon\sqrt{G^{-1}(\boldsymbol{\theta}^n)}\mathbf{z}^n \tag{4}$$

For a manifold whose metric tensor is globally constant, this reduces further to a pre-conditioned MALA proposal, where the pre-conditioning is effectively independent of the current parameter values. In this context, such pre-conditioning no longer needs to be chosen arbitrarily, but rather it may be informed by the geometry of the distribution we are exploring.

We point out that any approximations of the metric tensor would be best employed in the simplified mMALA scheme, defining the covariance of the proposal distribution, or as a flat approximation to a manifold. In the case of full mMALA, or even Hamiltonian Monte Carlo defined on a Riemannian manifold [1], Christoffel symbols are also used, incorporating the derivatives of the metric tensor as it changes across the surface of the manifold - in many cases the extra expense of computing or estimating such higher order information is not sufficiently supported by the increase in sampling efficiency [1] and for this reason we do not consider such methods further.

In the next section we consider the representation of the metric tensor as the covariance of the tangent vectors at each point. We consider a method of estimating this such that convergence is guaranteed by extending the state-space and introducing auxiliary variables that are conditioned on the current point and we demonstrate its potential within a Riemannian geometric context.

## 4 Approximate Geometry for MCMC Proposals

We first derive an acceptance ratio on an extended state-space that enables convergence to the stationary distribution before describing the implications for developing new differential geometric MCMC methods. Following [13, 14] we can employ the oft-used trick of defining an extended state space $\mathbf{X} \times \mathbf{D}$. We may of course choose $\mathbf{D}$ to be of any size, however in our particular case we shall choose $\mathbf{D}$ to be $\mathbb{R}^{m \times s}$, where $m$ is the dimension of the data and $s$ is the number of samples; the reasons for this shall become clear. We therefore sample from this extended state space, whose joint distribution follows as $\pi^* = \pi(\mathbf{x})\hat{\pi}(\mathbf{d}|\mathbf{x})$. Given the current states $[\mathbf{x_n}, \mathbf{d}_n]$, we may propose a new state $q(\mathbf{x}^*|\mathbf{x}_n, \mathbf{d}_n)$ and the MCMC algorithm will satisfy detailed balance and hence converge to the stationary distribution if we accept joint proposals with Metropolis-Hastings probability ratio,

$$
\begin{aligned}
\alpha(\mathbf{x}^*, \mathbf{d}^*|\mathbf{x}_n, \mathbf{d}_n) &= \min\left(1, \frac{\pi^*(\mathbf{x}^*, \mathbf{d}^*)}{\pi^*(\mathbf{x}_n, \mathbf{d}_n)}\frac{q(\mathbf{x}_n|\mathbf{x}^*, \mathbf{d}^*)}{q(\mathbf{x}^*|\mathbf{x}_n, \mathbf{d}_n)}\frac{\hat{\pi}(\mathbf{d}_n|\mathbf{x}_n)}{\hat{\pi}(\mathbf{d}^*|\mathbf{x}^*)}\right) \\
&= \min\left(1, \frac{\pi(\mathbf{x}^*)}{\pi(\mathbf{x}_n)}\frac{\hat{\pi}(\mathbf{d}^*|\mathbf{x}^*)}{\hat{\pi}(\mathbf{d}_n|\mathbf{x}_n)}\frac{q(\mathbf{x}_n|\mathbf{x}^*, \mathbf{d}^*)}{q(\mathbf{x}^*|\mathbf{x}_n, \mathbf{d}_n)}\frac{\hat{\pi}(\mathbf{d}_n|\mathbf{x}_n)}{\hat{\pi}(\mathbf{d}^*|\mathbf{x}^*)}\right) \\
&= \min\left(1, \frac{\pi(\mathbf{x}^*)}{\pi(\mathbf{x}_n)}\frac{q(\mathbf{x}_n|\mathbf{x}^*, \mathbf{d}^*)}{q(\mathbf{x}^*|\mathbf{x}_n, \mathbf{d}_n)}\right)
\end{aligned}
\tag{5}
$$

This is a reversible transition on $\pi(\mathbf{x}, \mathbf{d})$, from which we can sample to obtain $\pi(\mathbf{x})$ as the marginal distribution. The key point here is that we may define our proposal distribution $q(\mathbf{x}^*|\mathbf{x}_n, \mathbf{d}_n)$ in almost any deterministic manner we wish. In particular, choosing $\hat{\pi}(\mathbf{d}|\mathbf{x})$ to be the same distribution

as the log-likelihood for our statistical model, the $s$ samples from the extended state space $\mathbf{D}$ may be thought of as pseudo-data, from which we can deterministically calculate an estimate of the Expected Fisher Information to use as the covariance of a proposal distribution. Specifically, each sampled pseudo-data can be used deterministically to give a sample of $\frac{\partial \mathcal{L}}{\partial \boldsymbol{\theta}}$ given the current $\boldsymbol{\theta}$, all of which may then be used deterministically to obtain an approximation of the covariance of tangent vectors at the current point. This approximation, unlike the Hessian, will always be positive definite, and gives us an approximation of the metric tensor defining the local geometry. Further, we may use additional deterministic procedures, given $\mathbf{x}_n$ and $\mathbf{d}_n$, to construct better proposals; we consider a sparsity inducing approach in the next section.

## 5   Stability and Sparsity via $\ell_1$ Regularisation

We have two motivations for using an $\ell_1$ regularisation approach for computing the inverse of the metric tensor; firstly, since the metric is equivalent to the covariance of tangent vectors, we may obtain more stable estimates of the inverse metric tensor using smaller numbers of samples, and secondly, it induces a natural sparsity in the inverse metric, which may be exploited to decrease the computational cost associated with repeated Cholesky factorisations and matrix-vector multiplications. We adopted the graphical lasso [15, 16], in which the maximum likelihood solution results in the matrix optimisation problem,

$$\arg\min_{A \succ 0}\{-\log\det(A) + \operatorname{tr}(AG) + \gamma \sum_{i \neq j} |A_{ij}|\} \tag{6}$$

where $G$ is an empirical covariance matrix and $\gamma$ is a regularisation parameter. This convex optimisation problem aims to find $A$, the regularised maximum likelihood estimate for the inverse of the covariance matrix. Importantly, the optimisation algorithm we employ is deterministic given our tangent vectors, and therefore does not affect the validity of our MCMC algorithm; indeed we note that we may use any deterministic sparse matrix inverse estimation approaches within this MCMC algorithm. The use of the $\ell_1$ regularisation promotes sparsity [23]; larger values for the regularisation parameter matrix $\Lambda$ results in a solution that is more sparse, on the other hand when $\Lambda$ approaches zero, the solution converges to the inverse of $G$ (assuming it exists). It is also worth noting that the $\ell_1$ regularisation helps to recover a sparse structure in a high dimensional setting where the number of samples is less than the number of parameters [17].

In order to achieve sufficiently fast computation we carefully implemented the graphical lasso algorithm tailored to this problem. We used no penalisation for the diagonal and uniform regularisation parameter value for the off-diagonal elements. The motivation for not penalising the diagonal is that it has been shown in the covariance estimation setting that the true inverse is approached as the number of samples is increased [18], and the structure is learned more accurately [19]. The simple regularisation structure allowed code simplification and reduction in memory use. We refactored the graphical lasso algorithm of [15] and implemented it directly in FORTRAN which we then called from MATLAB, making sure to minimise matrix copying due to MATLAB processing. This code is available as a software package, GLASSOFAST [20].

In the current context, the use of this approach allows us to obtain sparse approximations to the inverse metric tensor, which may then be used in an MCMC proposal. Indeed, even if we have access to an analytic metric tensor we need not use the full inverse for our proposals; we could still obtain an approximate sparse representation, which may be beneficial computationally. The metric tensor varies smoothly across a Riemannian manifold and, theoretically, if we are calculating the inverse of 2 metric tensors that are close to each other, they may be numerically similar enough to be able to use the solution of one to speed up convergence of solution for the other, although in the simulations in this paper we found no benefit in doing so, i.e. the metric tensor varied too much as the MCMC sampler took large steps across the manifold.

## 6   Simulation Study

We consider a challenging class of statistical models that severely tests the sampling capability of MCMC methods; in particular, two examples based on nonlinear differential equations using a

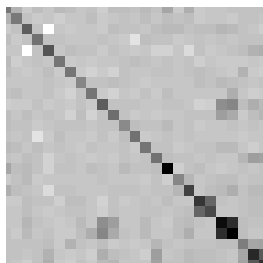
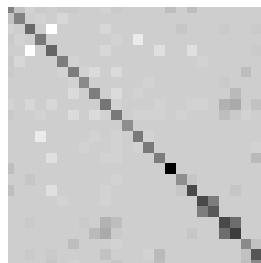

(a) Exact full inverse                    (b) Approximate sparse inverse

Figure 1: In this comparison we plotted the exact and the sparse approximate inverses of a typical metric tensor $G$; we note that only subsets of parameters are typically strongly correlated in the statistical models we consider here and that the sparse approximation still captures the main correlation structure present. Here the dimension is $p = 25$, and the regularisation parameter $\gamma$ is $0.05 \cdot ||G||_\infty$.

Table 1: Summary of results for the Fitzhugh-Nagumo model with 10 runs of each parameter sampling scheme and 5000 posterior samples.

| Sampling Method | Time (s) | Mean ESS $(a, b, c)$ | Total Time/ (Min mean ESS) | Relative Speed |
|---|---|---|---|---|
| Metropolis | 14.5 | 139, 18.2, 23.4 | 0.80 | $\times 1.1$ |
| MALA | 24.9 | 119.3, 28.7, 52.3 | 0.87 | $\times 1.0$ |
| mMALA Simp. | 35.9 | 283.4, 136.6, 173.7 | 0.26 | $\times 3.4$ |

biologically motivated robust Student-t likelihood, which renders the metric tensor analytically intractable. We examine the efficiency of our MCMC method with approximate metric on a well studied toy example, the Fitzhugh-Nagumo model, before examining a realistic, nonlinear and highly challenging example describing enzymatic circadian control in the plant Arabidopsis thaliana [22].

### 6.1 Nonlinear Ordinary Differential Equations

Statistical modelling using systems of nonlinear ordinary differential equations plays a vital role in unravelling the structure and behaviour of biological processes at a molecular level. The well-used Gaussian error model however is often inappropriate, particularly in molecular biology where limited measurements may not be repeated under exactly the same conditions and are susceptible to bias and systematic errors. The use of a Student-t distribution as a likelihood may help the robustness of the model with respect to possible outliers in the data. This presents a problem for standard manifold MCMC algorithms as it makes the metric tensor analytically intractable. We consider first the Fitzhugh-Nagumo model [1]. This synthetic dataset consisted of 200 time points simulated from the model between $t = [0, 20]$ with parameters $[a, b, c] = [0.2, 0.2, 3]$, to which Gaussian distributed noise was added with variance $\sigma^2 = 0.25$. We employed a Student-t likelihood with scaling parameter $v = 3$, and compared M-H and MALA (both employing scaled isotropic covariances), and simplified mMALA with approximate metric. The stepsize for each was automatically adjusted during the burn-in phase to obtain the theoretically optimal acceptance rate.

Table 1 shows the results including time-normalised effective sample size (ESS) as a measure of sampling efficiency excluding burn-in [1]. The approximate manifold sampler offers a modest improvement on the other two samplers; despite taking longer to run because of the computational cost of estimating the metric, the samples it draws exhibit lower autocorrelation, and as such the approximate manifold sampler offers the highest time-normalised ESS.

The toy Fitzhugh-Nagumo model is however rather simple, and despite being a popular example is rather unlike many realistic models used nowadays in the molecular modelling community. As such we consider another larger model that describes the enzymatic control of the circadian networks in Arabidopsis thaliana [21]. This is an extremely challenging, highly nonlinear model. We consider

Table 2: Comparison of pseudodata sample size on the quality of metric tensor estimation, and hence on sampling efficiency, using the circadian network example model, with 10 runs and 10,000 posterior samples.

| Number of Samples | Time (s) | Min Mean ESS | Total Time/ (Min mean ESS) | Relative Speed |
|---|---|---|---|---|
| 10 | 155.6 | 85.1 | 1.90 | ×1.0 |
| 20 | 163.2 | 171.9 | 0.95 | ×2.0 |
| 30 | 168.9 | 209.1 | 0.81 | ×2.35 |
| 40 | 175.2 | 208.3 | 0.84 | ×2.26 |

Table 3: Summary of results for the circadian network model with 10 runs of each parameter sampling scheme and 10,000 posterior samples.

| Sampling Method | Time (s) | Min Mean ESS | Total Time/ (Min mean ESS) | Relative Speed |
|---|---|---|---|---|
| Metropolis | 37.1 | 6.0 | 6.2 | ×4.4 |
| MALA | 101.3 | 3.7 | 27.4 | ×1.0 |
| Adaptive MCMC | 110.4 | 46.7 | 2.34 | ×11.7 |
| mMALA Simp. | 168.9 | 209.1 | 0.81 | ×33.8 |

inferring the 6 rate parameters that control production and decay of proteins in the nucleus and cytoplasm (see [22] for the equations and full details of the model), again employing a Student-t likelihood for which the Expected Fisher Information is analytically intractable. We used parameter values from [22] to simulate observations for each of the six species at 48 time points representing 48 hours in the model. Student-t distributed noise was then added to obtain the data for inference.

We first investigated the effect that the tangent vector sample size for covariance estimation has on the sampling efficiency of simplified mMALA. The results in Table 2 show that there is a threshold above which a more accurate estimate of the metric tensor does not result in additional sampling advantage. The threshold for this particular example model is around 30 pseudodata samples. Table 3 shows the time normalised statistical efficiency for each of the sampling methods; this time we also compare an Adaptive MCMC algorithm [2] with M-H, MALA, and simplified mMALA with approximate geometry. Both the M-H and MALA algorithms fail to explore the target distribution and have severe difficulties with the extreme scalings and nonlinear correlation structure present in the manifold. The Adaptive MCMC method works reasonably well after taking 2000 samples to learn the covariance structure, although its performance is still poorer than the simplified mMALA scheme, which converges almost immediately with no adaptation time required; the approximation mMALA makes of the local geometry allows it to adequately deal with the different scalings and correlations that occur in different parts of the space.

## 7  Conclusions

The use of Riemannian geometry can be very useful for enabling efficient sampling from arbitrary probability densities. The metric tensor may be used for creating position-specific proposal mechanisms that allow MCMC methods to automatically adapt to the local correlation structure induced by the sensitivities of the parameters of a statistical model. The metric tensor may conveniently be defined as the Expected Fisher Information, however this quantity is often either difficult or impossible to compute analytically. We have presented a sampling scheme that approximates the Expected Fisher Information by estimating the covariance structure of the tangent vectors at each point on the manifold. By considering this problem as one of inverse covariance estimation, this naturally led us to consider the use of $\ell_1$ regularisation to improve the estimation procedure. This had the added benefit of inducing sparsity into the metric tensor, which may offer computational advantages when proposing MCMC moves across the manifold. For future work it will be exciting to investigate the potential impact of approximate, sparse metric tensors for high dimensional problems.

Ben Calderhead gratefully acknowledges his Research Fellowship through the 2020 Science programme, funded by EPSRC grant number EP/I017909/1 and supported by Microsoft Research.

**References**

[1] M. Girolami and B. Calderhead, Riemann Manifold Langevin and Hamiltonian Monte Carlo Methods (with discussion), *Journal of the Royal Statistical Society: Series B*, 73:123-214, 2011

[2] H. Haario, E. Saksman and J. Tamminen, An Adaptive Metropolis Algorithm, *Bernoulli*, 7(2):223-242, 2001

[3] G. Roberts and J. Rosenthal, Examples of Adaptive MCMC, *Journal of Computational and Graphical Statistics*, 18(2), 2009

[4] G. Roberts and O. Stramer, Langevin diffusions and Metropolis-Hastings algorithms, *Methodol. Comput. Appl. Probab.*, 4, 337-358, 2003

[5] H. Tjelmeland and B. Hegstad, Mode Jumping Proposals in MCMC, *Scandinavian Journal of Statistics*, 28(1), 2001

[6] S. Amari and H. Nagaoka, Methods of Information Geometry, *Oxford University Press*, 2000

[7] Y. Qi and T. Minka, Hessian-based Markov Chain Monte-Carlo algorithms, *1st Cape Cod Workshop Monte Carlo Methods*, 2002

[8] A. Honkela, T. Raiko, M. Kuusela, M. Tornio and J. Karhunen, Approximate Riemannian conjugate gradient learning for fixed-form variational Bayes, *JMLR*, 11:3235-3268, 2010

[9] M. K. Murray and J. W. Rice, Differential Geometry and Statistics, *Chapman and Hall*, 1993

[10] C. R. Rao, Information and accuracy attainable in the estimation of statistical parameters, *Bull. Calc. Math. Soc.*, 37:81-91, 1945

[11] H. Jeffreys, Theory of Probability, *1st ed. The Clarendon Press, Oxford*, 1939

[12] J. Kent, Time reversible diffusions, *Adv. Appl. Probab.*, 10:819-835, 1978

[13] J. Besag, P. Green, D. Higdon, and K. Mengersen, Bayesian Computation and Stochastic Systems, *Statistical Science*, 10(1):3-41, 1995

[14] A. Doucet, P. Jacob and A. Johansen, Discussion of Riemann Manifold Langevin and Hamiltonian Monte Carlo Methods, *Journal of the Royal Statistical Society: Series B*, 73:162, 2011

[15] J. Friedman, T. Hastie and R. Tibshirani, Sparse inverse covariance estimation with the graphical lasso, *Biostatistics*, 9(3):432-441, 2008

[16] O. Banerjee, L. El Ghaoui and A. d'Aspremont, Model Selection Through Sparse Maximum Likelihood Estimation for Multivariate Gaussian or Binary Data, *JMLR*, 9(6), 2008

[17] P. Ravikumar, M. J. Wainwright, G. Raskutti, and B. Yu., Model selection in Gaussian graphical models: High-dimensional consistency of $\ell_1$-regularized MLE, *NIPS 21*, 2008

[18] A. J. Rothman, P. J. Bickel, E. Levina and J. Zhu, Sparse permutation invariant covariance estimation, *Electronic Journal of Statistics*, 2:494-515, 2008

[19] J. Duchi, S. Gould and D. Koller, Projected Subgradient Methods for Learning Sparse Gaussians, *Conference on Uncertainty in Artificial Intelligence*, 2008

[20] M. A. Sustik and B. Calderhead, GLASSOFAST: An efficient GLASSO implementation, *Technical Report*, Computer Science Department, University of Texas at Austin, TR-12-29, 2012

[21] J. C. W. Locke, A. Millar and M. Turner, Modelling genetic networks with noisy and varied experimental data: the circadian clock in Arabidopsis thaliana, *J. Theor. Biol.* 234:383-393, 2005

[22] B. Calderhead and M. Girolami, Statistical analysis of nonlinear dynamical systems using differential geometric sampling methods, *Journal of the Royal Society Interface Focus*, 1(6), 2011

[23] R. Tibshirani, Regression shrinkage and selection via the lasso, *Journal of the Royal Statistical Society: Series B*, 58:267-288, 1996

